# Discriminative Batch Mode Active Learning

**Yuhong Guo** and **Dale Schuurmans**
Department of Computing Science
University of Alberta
{yuhong, dale}@cs.ualberta.ca

## Abstract

Active learning sequentially selects unlabeled instances to label with the goal of reducing the effort needed to learn a good classifier. Most previous studies in active learning have focused on selecting one unlabeled instance to label at a time while retraining in each iteration. Recently a few batch mode active learning approaches have been proposed that select a *set* of most informative unlabeled instances in each iteration under the guidance of heuristic scores. In this paper, we propose a discriminative batch mode active learning approach that formulates the instance selection task as a continuous optimization problem over auxiliary instance selection variables. The optimization is formulated to maximize the discriminative classification performance of the target classifier, while also taking the unlabeled data into account. Although the objective is not convex, we can manipulate a quasi-Newton method to obtain a good local solution. Our empirical studies on UCI datasets show that the proposed active learning is more effective than current state-of-the art batch mode active learning algorithms.

## 1   Introduction

Learning a good classifier requires a sufficient number of labeled training instances. In many circumstances, unlabeled instances are easy to obtain, while labeling is expensive or time consuming. For example, it is easy to download a large number of webpages, however, it typically requires manual effort to produce classification labels for these pages. Randomly selecting unlabeled instances for labeling is inefficient in many situations, since non-informative or redundant instances might be selected. Hence, active learning (i.e., selective sampling) methods have been adopted to control the labeling process in many areas of machine learning, with the goal of reducing the overall labeling effort.

Given a large pool of unlabeled instances, active learning provides a way to iteratively select the most informative unlabeled instances—the queries—to label. This is the typical setting of pool-based active learning. Most active learning approaches, however, have focused on selecting only one unlabeled instance at one time, while retraining the classifier on each iteration. When the training process is hard or time consuming, this repeated retraining is inefficient. Furthermore, if a parallel labeling system is available, a single instance selection system can make wasteful use of the resource. Thus, a batch mode active learning strategy that selects multiple instances each time is more appropriate under these circumstances. Note that simply using a single instance selection strategy to select more than one unlabeled instance in each iteration does not work well, since it fails to take the information overlap between the multiple instances into account. Principles for batch mode active learning need to be developed to address the multi-instance selection specifically. In fact, a few batch mode active learning approaches have been proposed recently [2, 8, 9, 17, 19]. However, most extend existing single instance selection strategies into multi-instance selection simply by using a heuristic score or greedy procedure to ensure both the instance diversity and informativeness.

In this paper, we propose a new discriminative batch mode active learning strategy that exploits information from an unlabeled set to attempt to learn a good classifier directly. We define a good classifier to be one that obtains high likelihood on the labeled training instances and low uncertainty on labels of the unlabeled instances. We therefore formulate the instance selection problem as an optimization problem with respect to auxiliary instance selection variables, taking a combination of discriminative classification performance and label uncertainty as the objective function. Unfortunately, this optimization problem is NP-hard, thus seeking the optimal solution is intractable. However, we can approximate it locally using a second order Taylor expansion and obtain a suboptimal solution using a quasi-Newton local optimization technique.

The instance selection variables we introduce can be interpreted as indicating self-supervised, optimistic guesses for the labels of the selected unlabeled instances. A concern about the instance selection process, therefore, is that some information in the unlabeled data that is inconsistent with the true classification partition might mislead instance selection. Fortunately, the active learning method can immediately tell whether it has been misled, by comparing the true labels with its optimized guesses. Therefore, one can then adjust the active selection strategy to avoid such over-fitting in the next iteration, whenever a mismatch between the labeled and unlabeled data has been detected. An empirical study on UCI datasets shows that the proposed batch mode active learning method is more effective than some current state-of-the-art batch mode active learning algorithms.

## 2 Related Work

Many researchers have addressed the active learning problem in a variety of ways. Most have focused on selecting a single most informative unlabeled instance to label at a time. Many such approaches therefore make myopic decisions based solely on the current learned classifier, and select the unlabeled instance for which there is the greatest uncertainty. [10] chooses the unlabeled instance with conditional probability closest to 0.5 as the most uncertain instance. [5] takes the instance on which a committee of classifiers disagree the most. [3, 18] suggest choosing the instance closest to the classification boundary, where [18] analyzes this active learning strategy as a version space reduction process. Approaches that exploit unlabeled data to provide complementary information for active learning have also been proposed. [4, 20] exploit unlabeled data by using the prior density $p(\mathbf{x})$ as uncertainty weights. [16] selects the instance that optimizes the expected generalization error over the unlabeled data. [11] uses an EM approach to integrate information from unlabeled data. [13, 22] consider combining active learning with semi-supervised learning. [14] presents a mathematical model that explicitly combines clustering and active learning. [7] presents a discriminative approach that implicitly exploits the clustering information contained in the unlabeled data by considering optimistic labelings.

Since single instance selection strategies require tedious retraining with each instance labeled (and, moreover, since they cannot take advantage of parallel labeling systems), many batch mode active learning methods have recently been proposed. [2, 17, 19] extend single instance selection strategies that use support vector machines. [2] takes the diversity of the selected instances into account, in addition to individual informativeness. [19] proposes a representative sampling approach that selects the cluster centers of the instances lying within the margin of a support vector machine. [8, 9] choose multiple instances that efficiently reduce the Fisher information. Overall, these approaches use a variety of heuristics to guide the instance selection process, where the selected batch should be informative about the classification model while being diverse enough so that their information overlap is minimized.

Instead of using heuristic measures, in this paper, we formulate batch mode active learning as an optimization problem that aims to learn a good classifier directly. Our optimization selects the best set of unlabeled instances *and their labels* to produce a classifier that attains maximum likelihood on labels of the labeled instances while attaining minimum uncertainty on labels of the unlabeled instances. It is intractable to conduct an exhaustive search for the optimal solution; our optimization problem is NP-hard. Nevertheless we can exploit a second-order Taylor approximation and use a quasi-Newton optimization method to quickly reach a local solution. Our proposed approach provides an example of exploiting optimization techniques in batch model active learning research, much like other areas of machine learning where optimization techniques have been widely applied [1].

# 3 Logistic Regression

In this paper, we use binary logistic regression as the base classification algorithm. Logistic regression is a well-known and mature statistical model for probabilistic classification that has been actively studied and applied in machine learning. Given a test instance $\mathbf{x}$, binary logistic regression models the conditional probability of the class label $y \in \{+1, -1\}$ by

$$p(y|\mathbf{x}, \mathbf{w}) = \frac{1}{1 + \exp(-y\mathbf{w}^\top \mathbf{x})}$$

where $\mathbf{w}$ is the model parameter. Here the bias term is omitted for simplicity of notation. The model parameters can be trained by maximizing the likelihood of the labeled training data, i.e., minimizing the logloss of the training instances

$$\min_{\mathbf{w}} \sum_{i \in L} \log(1 + \exp(-y_i \mathbf{w}^\top \mathbf{x}_i)) + \frac{\lambda}{2}\mathbf{w}^\top \mathbf{w} \tag{1}$$

where $L$ indexes the training instances, and $\frac{\lambda}{2}\mathbf{w}^\top \mathbf{w}$ is a regularization term introduced to avoid over-fitting problems. Logistic regression is a robust classifier that can be trained efficiently using various convex optimization techniques [12]. Although it is a linear classifier, it is easy to obtain nonlinear classifications by simply introducing kernels [21].

# 4 Discriminative Batch Mode Active Learning

For active learning, one typically encounters a small number of labeled instances and a large number of unlabeled instances. Instance selection strategies based only on the labeled data therefore ignore potentially useful information embodied in the unlabeled instances. In this section, we present a new discriminative batch mode active learning algorithm for binary classification that exploits information in the unlabeled instances. The proposed approach is *discriminative* in the sense that (1) it selects a batch of instances by optimizing a discriminative classification model; and (2) it selects instances by considering the best discriminative configuration of their labels leading to the best classifier. Unlike other batch mode active learning methods, which identify the most informative batch of instances using heuristic measures, our approach aims to identify the batch of instances that directly optimizes classification performance.

## 4.1 Optimization Problem

An optimal active learning strategy selects a set of instances to label that leads to learning the best classifier. We assume the learner selects a set of a fixed size $m$, which is chosen as a parameter. Supervised learning methods typically maximize the likelihood of training instances. With unlabeled data being available, semi-supervised learning methods have been proposed that train by simultaneously maximizing the likelihood of labeled instances and minimizing the uncertainty of the labels for unlabeled instances [6]. That is, to achieve a classifier with better generalization performance, one can maximizing the expected log likelihood of the labeled data and minimize the entropy of the missing labels on the unlabeled data, according to

$$\sum_{i \in L} \log P(y_i|\mathbf{x}_i, \mathbf{w}) + \alpha \sum_{j \in U} \sum_{y = \pm 1} P(y|\mathbf{x}_j, \mathbf{w}) \log P(y|\mathbf{x}_j, \mathbf{w}) \tag{2}$$

where $\alpha$ is a tradeoff parameter used to adjust the relative influence of the labeled and unlabeled data, $\mathbf{w}$ specifies the conditional model, $L$ indexes the labeled instances, and $U$ indexes the unlabeled instances.

The new active learning approach we propose is motivated by this semi-supervised learning principle. We propose to select a batch of $m$ unlabeled instances, $S$, to label in each iteration from the total unlabeled set $U$, with the goal of maximizing the objective (2). Specifically, we define the score function for a set of selected instances $S$ in iteration $t + 1$ as follows

$$f(S) = \sum_{i \in L^t \cup S} \log P(y_i|\mathbf{x}_i, \mathbf{w}^{t+1}) - \alpha \sum_{j \in U^t \setminus S} H(y|\mathbf{x}_j, \mathbf{w}^{t+1}) \tag{3}$$

where $\mathbf{w}^{t+1}$ is the parameter set for the conditional classification model trained on the new labeled set $L^{t+1} = L^t \cup S$, and $H(y|\mathbf{x}_j, \mathbf{w}^{t+1})$ denotes the entropy of the conditional distribution $P(y|\mathbf{x}_j, \mathbf{w}^{t+1})$, such that

$$H(y|\mathbf{x}_j, \mathbf{w}^{t+1}) = -\sum_{y=\pm 1} P(y|\mathbf{x}_j, \mathbf{w}^{t+1}) \log P(y|\mathbf{x}_j, \mathbf{w}^{t+1})$$

The proposed active learning strategy is to select the batch of instances that has the highest score.

In practice, however it is problematic to use the $f(S)$ score directly to guide instance selection: the labels for instances $S$ are not known when the selection is conducted. One typical solution for this problem is to use the expected $f(S)$ score computed under the current conditional model specified by $\mathbf{w}^t$

$$\mathbf{E}[f(S)] = \sum_{\mathbf{y}_S} P(\mathbf{y}_S|\mathbf{x}_S, \mathbf{w}^t) f(S)$$

However, using $P(\mathbf{y}_S|\mathbf{x}_S, \mathbf{w}^t)$ as weights, this expectation might aggravate any ambiguity that already exists in the current classification model $\mathbf{w}^t$, since it has been trained on a very small labeled set $L^t$. Instead, we propose an optimistic strategy: use the best $f(S)$ score that the batch of unlabeled instances $S$ can achieve over all possible label configurations. This optimistic scoring function can be written as

$$f(S) = \max_{\mathbf{y}_S} \sum_{i \in L^t \cup S} \log P(y_i|\mathbf{x}_i, \mathbf{w}^{t+1}) - \alpha \sum_{j \in U^t \setminus S} H(y|\mathbf{x}_j, \mathbf{w}^{t+1}) \qquad (4)$$

Thus the problem becomes how to select a set of instances $S$ that achieves the best optimistic $f(S)$ score defined in (4). Although this problem can be solved using an exhaustive search on all size $m$ subsets, $S$, of the unlabeled set $U$, it is intractable to do so in practice since the search space is exponentially large. Explicit heuristic search approaches seeking a local optima do not exist either, since it is hard to define an efficient set of operators that can transfer from one position to another one within the search space while guaranteeing improvements to the optimistic score.

Instead, in this paper we propose to approach the problem by formulating optimistic batch mode active learning as an explicit mathematical optimization. Given the labeled set $L^t$ and unlabeled set $U^t$ after iteration $t$, the task in iteration $t + 1$ is to select a size $m$ subset $S$ from $U^t$ that achieves the best score defined in (4). To do so, we first introduce a set of $\{0, 1\}$-valued instance selection variables $\boldsymbol{\mu}$. In particular, $\boldsymbol{\mu}$ is a $|U^t| \times 2$ sized indicator matrix, where each row vector $\boldsymbol{\mu}_j$ corresponds to the two possible labels $\{+1, -1\}$ of the $j$th instance in $U^t$. Then the optimistic instance selection for iteration $t + 1$ can be formulated as the following optimization problem

$$\max_{\boldsymbol{\mu}} \quad \sum_{i \in L^t} \log P(y_i|\mathbf{x}_i, \mathbf{w}^{t+1}) + \beta \sum_{j \in U^t} \mathbf{v}_j^{t+1} \boldsymbol{\mu}_j^\top - \alpha \sum_{j \in U^t} (1 - \boldsymbol{\mu}_j \mathbf{e}) H(y|\mathbf{x}_j, \mathbf{w}^{t+1}) \quad (5)$$

$$s.t. \quad \boldsymbol{\mu} \in \{0, 1\}^{|U^t| \times 2} \qquad (6)$$

$$\boldsymbol{\mu} \bullet E = m \qquad (7)$$

$$\boldsymbol{\mu}_j \mathbf{e} \leq 1, \forall j \qquad (8)$$

$$\mathbf{1}^\top \boldsymbol{\mu} \leq \left(\frac{1}{2} + \epsilon\right) m \mathbf{e}^\top \qquad (9)$$

where $\mathbf{v}_j^{t+1}$ is a row vector $[\log P(y = 1|\mathbf{x}_j, \mathbf{w}^{t+1}), \log P(y = -1|\mathbf{x}_j, \mathbf{w}^{t+1})]$; $\mathbf{e}$ is a 2-entry column vector with all 1s; $\mathbf{1}$ is a $|U^t|$-entry column vector with all 1s; $E$ is a $U^t \times 2$ sized matrix with all 1s; $\bullet$ is matrix inner product; $\epsilon$ is a user-provided parameter that controls class balance during instance selection; and $\beta$ is a parameter that we will use later to adjust our belief in the guessed labels. Note that, the selection variables $\boldsymbol{\mu}$ not only choose instances from $U^t$, but also select labels for the selected instances. Solving this optimization yields the optimal $\boldsymbol{\mu}$ for instance selection in iteration $t + 1$.

The optimization problem (5) is an integer programming problem that produces equivalent results to using exhaustive search to optimize (4), except that we have additional class balance constraints (9). Integer programming is an NP-hard problem. Thus, the first step toward solving this problem in practice is to relax it into a continuous optimization by replacing the integer constraints (6) with

continuous constraints $0 \leq \boldsymbol{\mu} \leq 1$, yielding the relaxed formulation

$$\max_{\boldsymbol{\mu}} \quad \sum_{i \in L^t} \log P(y_i|\mathbf{x}_i, \mathbf{w}^{t+1}) + \beta \sum_{j \in U^t} \mathbf{v}_j^{t+1} \boldsymbol{\mu}_j^\top - \alpha \sum_{j \in U^t} (1 - \boldsymbol{\mu}_j \mathbf{e}) H(y|\mathbf{x}_j, \mathbf{w}^{t+1}) \quad (10)$$

$$s.t. \quad 0 \leq \boldsymbol{\mu} \leq 1 \quad (11)$$

$$\boldsymbol{\mu} \bullet E = m \quad (12)$$

$$\boldsymbol{\mu}_j \mathbf{e} \leq 1, \forall j \quad (13)$$

$$\mathbf{1}^\top \boldsymbol{\mu} \leq \left(\frac{1}{2} + \epsilon\right) m \mathbf{e}^\top \quad (14)$$

If we can solve this continuous optimization problem, a greedy strategy can then be used to recover the integer solution by iteratively setting the largest non-integer $\boldsymbol{\mu}$ value to 1 with respect to the constraints. However, this relaxed optimization problem is still very complex: the objective function (10) is not a concave function of $\boldsymbol{\mu}$.[1] Nevertheless, standard continuous optimization techniques can be used to solve for a local maxima.

## 4.2 Quasi-Newton Method

To derive a local optimization technique, consider the objective function (10) as a function of the instance selection variables $\boldsymbol{\mu}$

$$f(\boldsymbol{\mu}) = \sum_{i \in L^t} \log P(y_i|\mathbf{x}_i, \mathbf{w}^{t+1}) + \beta \sum_{j \in U^t} \mathbf{v}_j^{t+1} \boldsymbol{\mu}_j^\top - \alpha \sum_{j \in U^t} (1 - \boldsymbol{\mu}_j \mathbf{e}) H(y|\mathbf{x}_j, \mathbf{w}^{t+1}) \quad (15)$$

As noted, this function is non-concave, therefore convenient convex optimization techniques that achieve global optimal solutions cannot be applied. Nevertheless, a local optimization approach exploiting quasi-Newton methods can quickly determine a local optimal solution $\boldsymbol{\mu}^*$. Such a local optimization approach iteratively updates $\boldsymbol{\mu}$ to improve the objective (15), and stops when a local maximum is reached. At each iteration, it makes a local move that allows it to achieve the largest improvement in the objective function along the direction decided by cumulative information obtained from the sequence of local gradients. Suppose $\bar{\boldsymbol{\mu}}_{(k)}$ is the starting point for iteration $k$. We first derive a second-order Taylor approximation $\tilde{f}(\boldsymbol{\mu})$ for the objective function $f(\boldsymbol{\mu})$ at $\bar{\boldsymbol{\mu}}_{(k)}$

$$\tilde{f}(\boldsymbol{\mu}) = f(\bar{\boldsymbol{\mu}}_{(k)}) + \nabla f_k^\top \text{vec}(\boldsymbol{\mu} - \bar{\boldsymbol{\mu}}_{(k)}) + \frac{1}{2}\text{vec}(\boldsymbol{\mu} - \bar{\boldsymbol{\mu}}_{(k)})^\top H_k \text{ vec}(\boldsymbol{\mu} - \bar{\boldsymbol{\mu}}_{(k)}) \quad (16)$$

where $\text{vec}(\cdot)$ is a function that transforms a matrix into a column vector, and $\nabla f_k = \nabla f(\bar{\boldsymbol{\mu}}_{(k)})$ and $H_k$ denote the gradient vector and Hessian matrix of $f(\boldsymbol{\mu})$ at point $\bar{\boldsymbol{\mu}}_{(k)}$, respectively. Since our original optimization function $f(\boldsymbol{\mu})$ is smooth, the quadratic function $\tilde{f}(\boldsymbol{\mu})$ can reasonably approximate it in a small neighborhood of $\bar{\boldsymbol{\mu}}_{(k)}$. Thus we can determine our update direction by solving a quadratic programming with the objective (16) and linear constraints (11), (12), (13) and (14). Suppose the optimal solution for this quadratic program is $\tilde{\boldsymbol{\mu}}_{(k)}^*$. Then a reasonable update direction $\mathbf{d}_k = \tilde{\boldsymbol{\mu}}_{(k)}^* - \bar{\boldsymbol{\mu}}_{(k)}$ can be obtained for iteration $k$. Given this direction, a backtrack line search can be used to guarantee improvement over the original objective (15). Note that for each different value of $\boldsymbol{\mu}$, $\mathbf{w}^{t+1}$ has to be retrained on $L^t \cup S$ to evaluate the new objective value, since $S$ is determined by $\boldsymbol{\mu}$. In order to reduce the computational cost, we approximate the training of $\mathbf{w}^{t+1}$ in our empirical study, by limiting it to a few Newton-steps with a starting point given by $\mathbf{w}^t$ trained only on $L^t$.

The remaining issue is to compute the local gradient $\nabla f(\bar{\boldsymbol{\mu}}_{(k)})$ and the Hessian matrix $H_k$. We assume $\mathbf{w}^{t+1}$ remains constant with small local updates on $\bar{\boldsymbol{\mu}}$. Thus the local gradient can be approximated as

$$\nabla f(\bar{\boldsymbol{\mu}}_{j(k)}) = \beta \mathbf{v}_j^{t+1} + \alpha \left[ H(y|\mathbf{x}_j, \mathbf{w}^{t+1}), H(y|\mathbf{x}_j, \mathbf{w}^{t+1}) \right]$$

and therefore $\nabla f(\bar{\boldsymbol{\mu}}_{(k)})$ can be constructed from the individual $\nabla f(\bar{\boldsymbol{\mu}}_{j(k)})$. We then use BFGS (Broyden-Fletcher-Goldfarb-Shanno) to compute the Hessian matrix, which starts as an identity matrix for the first iteration, and is updated in each iteration as follows [15]

$$H_{k+1} = H_k - \frac{H_k s_k s_k^\top H_k}{s_k^\top H_k s_k} + \frac{y_k y_k^\top}{y_k^\top s_k}$$

where $y_k = \nabla f_{k+1} - \nabla f_k$, and $s_k = \bar{\boldsymbol{\mu}}_{(k+1)} - \bar{\boldsymbol{\mu}}_{(k)}$. This Hessian matrix accumulates information from the sequences of local gradients to help determine better update directions.

## 4.3 Adjustment Strategy

In the discriminative optimization problem formulated in Section 4.1, the $\boldsymbol{\mu}$ variables are used to optimistically select both instances and their labels, with the goal of achieving the best classification model according to the objective (5). However, when the labeled set is small and the discriminative partition (clustering) information contained in the large unlabeled set is inconsistent with the true classification, the labels optimistically *guessed* for the selected instances through $\boldsymbol{\mu}$ might not match the underlying true labels. When this occurs, the instance selected will not be very useful for identifying the true classification model. Furthermore, the unlabeled data might continue to mislead the next instance selection iteration.

Fortunately, we can immediately identify when the process has been misled once the true labels for the selected instances have been obtained. If the true labels are different from the labels guessed by the optimization, we need to make an adjustment for the next instance selection iteration. We have tried a few adjustment strategies in our study, but report the most effective one in this paper. Note that the *being-misled* problem is caused by the unlabeled data, which affects the target classification model through the term $\beta \sum_{j \in U^t} \mathbf{v}_j^{t+1} \boldsymbol{\mu}_j^{\top}$. Therefore, a simple way to fix the problem is to adjust the parameter $\beta$. Specifically, at the end of each iteration $t$, we obtain the true labels $\mathbf{y}_S$ for the selected instances $S$, and compare them with our guessed labels $\hat{\mathbf{y}}_S$ indicated by $\boldsymbol{\mu}^*$. If they are consistent, we will set $\beta = 1$, which means we trust the partition information from the unlabeled data as same as the label information in the labeled data for building the classification model. If $\mathbf{y}_S \neq \hat{\mathbf{y}}_S$, apparently we should reduce the $\beta$ value, that is, reducing the influence of the unlabeled data for the next selection iteration $t + 1$. We use a simple heuristic procedure to determine the $\beta$ value in this case. Starting from $\beta = 1$, we then multiplicatively reduce its value by a small factor, 0.5, until a better objective value for (15) can be obtained when replacing the guessed indicator variables $\boldsymbol{\mu}^*$ with the true label indicators. Note that, if we reduce $\beta$ to zero, our optimization problem will be exactly equivalent to picking the most uncertain instance (when $m = 1$).

## 5 Experiments

To investigate the empirical performance of the proposed discriminative batch mode active learning algorithm (*Discriminative*), we conducted a set of experiments on nine two-class UCI datasets, comparing with a baseline random instance selection algorithm (*Random*), a non-batch myopic active learning method that selects the most uncertain instance each time (*MostUncertain*), and two batch mode active learning methods proposed in the literature: *svmD*, an approach that incorporates diversity in active learning with SVM [2]; and *Fisher*, an approach that uses Fisher information matrix for instance selection [9]. The UCI datasets we used include (we show the name, followed by the number of instances and the number of attributes): Australian(690;14), Cleve(303;13), Corral(128;6), Crx(690;15), Flare(1066;10), Glass2(163;9), Heart(270;13), Hepatitis(155;20) and Vote(435;15).

We consider a hard case of active learning, where only a few labeled instances are given at the start. In each experiment, we start with four randomly selected labeled instances, two in each class. We then randomly select 2/3 of the remaining instances as the unlabeled set, using the remaining instances for testing. All the algorithms start with the same initial labeled set, unlabeled set and testing set. For a fixed batch size $m$, each algorithm repeatedly select $m$ instances to label each time. In this section, we report the experimental results with $m = 5$, averaged over 20 times repetitions.

Figure 1 shows the comparison results on the nine UCI datasets. These results suggest that although the baseline random sampling method, *Random*, works surprisingly well in our experiments, the proposed algorithm, *Discriminative*, always performs better or at least achieves a comparable performance. Moreover, *Discriminative* also apparently outperforms the other two batch mode algorithms, *svmD* and *Fisher*, on five datasets—Australian, Cleve, Flare, Heart and Hepatitis, and reaches a tie on two datasets—Crx and Vote. The myopic most uncertain selection method, *MostUncertain*, shows an overall inferior performance to *Discriminative* on Australian, Cleve, Crx, Heart and Hepatitis, and achieves a tie on Flare and Vote. However, *Discriminative* demonstrates weak perfor-

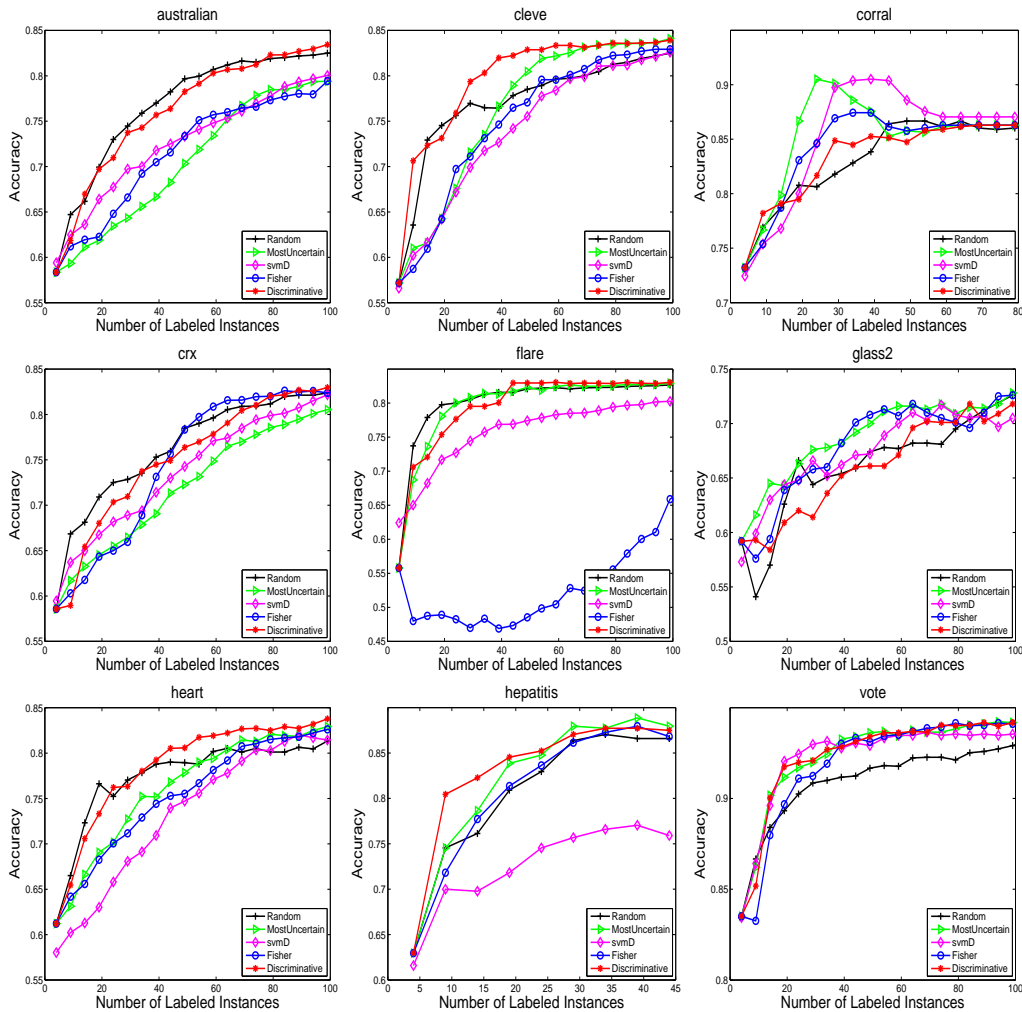

Figure 1: Results on UCI Datasets

mance on two datasets—Corral and Glass2, where the evaluation lines for most algorithms in the figures are strangely very bumpy. The reason behind this remains to be investigated.

These empirical results suggest that selecting unlabeled instances through optimizing the classification model directly would obtain more relevant and informative instances, comparing with using heuristic scores to guide the selection. Although the original optimization problem formulated is NP-hard, a relaxed local optimization method that leads to a local optimal solution still works effectively.

## 6   Conclusion

In this paper, we proposed a discriminative batch mode active learning approach that exploits information in unlabeled data and selects a batch of instances by optimizing the target classification model. Although the proposed technique could be overly optimistic about the information presented by the unlabeled set, and consequently be misled, this problem can be identified immediately after obtaining the true labels. A simple adjustment strategy can then be used to rectify the problem in the following iteration. Experimental results on UCI datasets show that this approach is generally more effective comparing with other batch mode active learning methods, a random sampling method, and a myopic non-batch mode active learning method. Our current work is focused on 2-class classification problems, however, it is easy to be extended to multiclass classification problems.

## Footnotes

[1]Note that $\mathbf{w}^{t+1}$ is the classification model parameter set trained on $L^{t+1} = L^t \cup S$, where $S$ indexes the unlabeled instances selected by $\boldsymbol{\mu}$. Therefore $\mathbf{w}^{t+1}$ is a function of $\boldsymbol{\mu}$.

# References

[1] K. Bennett and E. Parrado-Hernandez. The interplay of optimization and machine learning research. *Journal of Machine Learning Research*, 7, 2006.

[2] K. Brinker. Incorporating diversity in active learning with support vector machines. In *Proceedings of the 20th International Conference on Machine learning*, 2003.

[3] C. Campbell, N. Cristianini, and A. Smola. Query learning with large margin classifiers. In *Proceedings of the 17th International Conference on Machine Learning*, 2000.

[4] D. Cohn, Z. Ghahramani, and M. Jordan. Active learning with statistical models. *Journal of Artificial Intelligence Research*, 4, 1996.

[5] Y. Freund, H. S. Seung, E. Shamir, and N. Tishby. Selective sampling using the query by committee algorithm. *Machine Learning*, 28, 1997.

[6] Y. Grandvalet and Y. Bengio. Semi-supervised learning by entropy minimization. In *Advances in Neural Information Processing Systems*, 2005.

[7] Y. Guo and R. Greiner. Optimistic active learning using mutual information. In *Proceedings of the International Joint Conference on Artificial Intelligence*, 2007.

[8] S. Hoi, R. Jin, and M. Lyu. Large-scale text categorization by batch mode active learning. In *Proceedings of the International World Wide Web Conference*, 2006.

[9] S. Hoi, R. Jin, J. Zhu, and M. Lyu. Batch mode active learning and its application to medical image classification. In *Proceedings of the 23rd International Conference on Machine Learning*, 2006.

[10] D. Lewis and W. Gale. A sequential algorithm for training text classifiers. In *Proceedings of the International ACM-SIGIR Conference on Research and Development in Information Retrieval*, 1994.

[11] A. McCallum and K. Nigam. Employing EM in pool-based active learning for text classification. In *Proceedings of the 15th International Conference on Machine Learning*, 1998.

[12] T. Minka. A comparison of numerical optimizers for logistic regression. Technical report, 2003. http://research.microsoft.com/ minka/papers/logreg/.

[13] I. Muslea, S. Minton, and C. Knoblock. Active + semi-supervised learning = robust multi-view learning. In *Proceedings of the 19th International Conference on Machine Learning*, 2002.

[14] H. Nguyen and A. Smeulders. Active learning using pre-clustering. In *Proceedings of the 21st International Conference on Machine Learning*, 2004.

[15] J. Nocedal and S.J. Wright. *Numerical Optimization*. Springer, New York, 1999.

[16] N. Roy and A. McCallum. Toward optimal active learning through sampling estimation of error reduction. In *Proceedings of the 18th International Conference on Machine Learning*, 2001.

[17] G. Schohn and D. Cohn. Less is more: Active learning with support vector machines. In *Proceedings of the 17th International Conference on Machine Learning*, 2000.

[18] S. Tong and D. Koller. Support vector machine active learning with applications to text classification. In *Proceedings of the 17th International Conference on Machine Learning*, 2000.

[19] Z. Xu, K. Yu, V. Tresp, X. Xu, and J. Wang. Representative sampling for text classification using support vector machines. In *Proceedings of the 25th European Conference on Information Retrieval Research*, 2003.

[20] C. Zhang and T. Chen. An active learning framework for content-based information retrieval. *IEEE Trans on Multimedia*, 4:260–258, 2002.

[21] J. Zhu and T. Hastie. Kernel logistic regression and the import vector machine. *Journal of Computational and Graphical Statistics*, 14, 2005.

[22] X. Zhu, J. Lafferty, and Z. Ghahramani. Combining active learning and semi-supervised learning using gaussian fields and harmonic functions. In *ICML Workshop on The Continuum from Labeled to Unlabeled Data in Machine Learning and Data Mining*, 2003.
